# Kernelized Sorting

**Novi Quadrianto**
RSISE, ANU & SML, NICTA
Canberra, ACT, Australia
novi.quad@gmail.com

**Le Song**
SCS, CMU
Pittsburgh, PA, USA
lesong@cs.cmu.edu

**Alex J. Smola**
Yahoo! Research
Santa Clara, CA, USA
alex@smola.org

## Abstract

Object matching is a fundamental operation in data analysis. It typically requires the definition of a similarity measure *between* the classes of objects to be matched. Instead, we develop an approach which is able to perform matching by requiring a similarity measure only *within* each of the classes. This is achieved by maximizing the dependency between matched pairs of observations by means of the Hilbert Schmidt Independence Criterion. This problem can be cast as one of maximizing a quadratic assignment problem with special structure and we present a simple algorithm for finding a locally optimal solution.

## 1 Introduction

Matching pairs of objects is a fundamental operation of unsupervised learning. For instance, we might want to match a photo with a textual description of a person, a map with a satellite image, or a music score with a music performance. In those cases it is desirable to have a compatibility function which determines how one set may be translated into the other. For many such instances we may be able to *design* a compatibility score based on prior knowledge or to observe one based on the co-occurrence of such objects.

In some cases, however, such a match may not exist or it may not be given to us beforehand. That is, while we may have a good understanding of two sources of observations, say $\mathcal{X}$ and $\mathcal{Y}$, we may not understand the mapping between the two spaces. For instance, we might have two collections of documents purportedly covering the same content, written in two different languages. Here it should be our goal to determine the correspondence between both sets and to identify a mapping between the two domains. In the following we present a method which is able to perform such matching *without* the need of a cross-domain similarity measure.

Our method relies on the fact that one may estimate the *dependence* between sets of random variables even without knowing the cross-domain mapping. Various criteria are available. We choose the Hilbert Schmidt Independence Criterion between two sets and we maximize over the permutation group to find a good match. As a side-effect we obtain an explicit representation of the covariance.

We show that our method generalizes sorting. When using a different measure of dependence, namely an approximation of the mutual information, our method is related to an algorithm of [1]. Finally, we give a simple approximation algorithm for kernelized sorting.

### 1.1 Sorting and Matching

The basic idea underlying our algorithm is simple. Denote by $X = \{x_1, \ldots, x_m\} \subseteq \mathcal{X}$ and $Y = \{y_1, \ldots, y_m\} \subseteq \mathcal{Y}$ two sets of observations between which we would like to find a correspondence. That is, we would like to find some permutation $\pi \in \Pi_m$ on $m$ terms, that is

$$\Pi_m := \left\{ \pi | \pi \in \{0, 1\}^{m \times m} \text{ and } \pi 1_m = 1_m \text{ and } \pi^\top 1_m = 1_m \right\}, \tag{1}$$

such that the pairs $Z(\pi) := \left\{ (x_i, y_{\pi(i)}) \text{ for } 1 \le i \le m \right\}$ correspond to dependent random variables. Here $1_m \in \mathbb{R}^m$ is the vector of all ones. We seek a permutation $\pi$ such that the mapping $x_i \to y_{\pi(i)}$ and its converse mapping from $y$ to $x$ are simple. Denote by $D(Z(\pi))$ a measure of the dependence between $x$ and $y$. Then we define nonparametric sorting of $X$ and $Y$ as follows

$$\pi^* := \operatorname{argmax}_{\pi \in \Pi_m} D(Z(\pi)). \tag{2}$$

This paper is concerned with measures of $D$ and approximate algorithms for (2). In particular we will investigate the Hilbert Schmidt Independence Criterion and the Mutual Information.

## 2  Hilbert Schmidt Independence Criterion

Let sets of observations $X$ and $Y$ be drawn jointly from some probability distribution $\operatorname{Pr}_{xy}$. The Hilbert Schmidt Independence Criterion (HSIC) [2] measures the dependence between $x$ and $y$ by computing the norm of the cross-covariance operator over the domain $\mathcal{X} \times \mathcal{Y}$ in Hilbert Space. It can be shown, provided the Hilbert Space is universal, that this norm vanishes if and only if $x$ and $y$ are independent. A large value suggests strong dependence with respect to the choice of kernels.

Formally, let $\mathcal{F}$ be the Reproducing Kernel Hilbert Space (RKHS) on $\mathcal{X}$ with associated kernel $k : \mathcal{X} \times \mathcal{X} \to \mathbb{R}$ and feature map $\phi : \mathcal{X} \to \mathcal{F}$. Let $\mathcal{G}$ be the RKHS on $\mathcal{Y}$ with kernel $l$ and feature map $\psi$. The cross-covariance operator $\mathcal{C}_{xy} : \mathcal{G} \mapsto \mathcal{F}$ is defined by [3] as

$$\mathcal{C}_{xy} = \mathbf{E}_{xy}[(\phi(x) - \mu_x) \otimes (\psi(y) - \mu_y)], \tag{3}$$

where $\mu_x = \mathbf{E}[\phi(x)]$, $\mu_y = \mathbf{E}[\psi(y)]$, and $\otimes$ is the tensor product. HSIC, denoted as $\mathcal{D}$, is then defined as the square of the Hilbert-Schmidt norm of $\mathcal{C}_{xy}$ [2] via $\mathcal{D}(\mathcal{F}, \mathcal{G}, \operatorname{Pr}_{xy}) := \|\mathcal{C}_{xy}\|_{\mathrm{HS}}^2$. In term of kernels HSIC can be expressed as

$$\mathbf{E}_{xx'yy'}[k(x, x')l(y, y')] + \mathbf{E}_{xx'}[k(x, x')]\mathbf{E}_{yy'}[l(y, y')] - 2\mathbf{E}_{xy}[\mathbf{E}_{x'}[k(x, x')]\mathbf{E}_{y'}[l(y, y')]], \tag{4}$$

where $\mathbf{E}_{xx'yy'}$ is the expectation over both $(x, y) \sim \operatorname{Pr}_{xy}$ and an additional pair of variables $(x', y') \sim \operatorname{Pr}_{xy}$ drawn *independently* according to the same law. Given a sample $Z = \{(x_1, y_1), \ldots, (x_m, y_m)\}$ of size $m$ drawn from $\operatorname{Pr}_{xy}$ an empirical estimate of HSIC is

$$D(\mathcal{F}, \mathcal{G}, Z) = (m-1)^{-2} \operatorname{tr} HKHL = (m-1)^{-2} \operatorname{tr} \bar{K}\bar{L}. \tag{5}$$

where $K, L \in \mathbb{R}^{m \times m}$ are the kernel matrices for the data and the labels respectively, i.e. $K_{ij} = k(x_i, x_j)$ and $L_{ij} = l(y_i, y_j)$. Moreover, $H_{ij} = \delta_{ij} - m^{-1}$ centers the data and the labels in feature space. Finally, $\bar{K} := HKH$ and $\bar{L} := HLH$ denote the centered versions of $K$ and $L$ respectively. Note that (5) is a *biased* estimate where the expectations with respect to $x, x', y, y'$ have all been replaced by empirical averages over the set of observations.

### 2.1  Kernelized Sorting

Previous work used HSIC to *measure* independence between given random variables [2]. Here we use it to *construct* a mapping between $X$ and $Y$ by permuting $Y$ to maximize dependence. There are several advantages in using HSIC as a dependence criterion. First, HSIC satisfies concentration of measure conditions [2]. That is, for random draws of observation from $\operatorname{Pr}_{xy}$, HSIC provides values which are very similar. This is desirable, as we want our mapping to be robust to small changes. Second, HSIC is easy to compute, since only the kernel matrices are required and no density estimation is needed. The freedom of choosing a kernel allows us to incorporate prior knowledge into the dependence estimation process. The consequence is that we are able to generate a family of methods by simply choosing appropriate kernels for $X$ and $Y$.

**Lemma 1** *The nonparametric sorting problem is given by* $\pi^* = \operatorname{argmax}_{\pi \in \Pi_m} \operatorname{tr} \bar{K}\pi^\top \bar{L}\pi$.

**Proof**  We only need to establish that $H\pi = \pi H$ since the rest follows from the definition of (5). Note that since $H$ is a centering matrix, it has the eigenvalue 0 for the vector of all ones and the eigenvalue 1 for all vectors orthogonal to that. Next note that the vector of all ones is also an eigenvector of any permutation matrix $\pi$ with $\pi 1 = 1$. Hence $H$ and $\pi$ matrices commute. ∎

Next we show that the objective function is indeed reasonable: for this we need the following inequality due to Polya, Littlewood and Hardy:

**Lemma 2** *Let $a, b \in \mathbb{R}^m$ where $a$ is sorted ascendingly. Then $a^\top \pi b$ is maximized for $\pi = \operatorname{argsort} b$.*

**Lemma 3** *Let $\mathfrak{X} = \mathcal{Y} = \mathbb{R}$ and let $k(x, x') = xx'$ and $l(y, y') = yy'$. Moreover, assume that $x$ is sorted ascendingly. In this case (5) is maximized by either $\pi = \operatorname{argsort} y$ or by $\pi = \operatorname{argsort} -y$.*

**Proof** Under the assumptions we have that $\bar{K} = Hxx^\top H$ and $\bar{L} = Hyy^\top H$. Hence we may rewrite the objective as $\left[(Hx)^\top \pi (Hy)\right]^2$. This is maximized by sorting $Hy$ ascendingly. Since the centering matrix $H$ only changes the offset but not the order this is equivalent to sorting $y$. We have two alternatives, since the objective function is insensitive to sign reversal of $y$. ∎

This means that sorting is a special case of kernelized sorting, hence the name. In fact, when solving the general problem, it turns out that a projection onto the principal eigenvectors of $\bar{K}$ and $\bar{L}$ is a good initialization of an optimization procedure.

## 2.2 Diagonal Dominance

In some cases the biased estimate of HSIC as given in (5) leads to very undesirable results, in particular in the case of document analysis. This is the case since kernel matrices on texts tend to be diagonally dominant: a document tends to be *much* more similar to itself than to others. In this case the $O(1/m)$ bias of (5) is significant. Unfortunately, the minimum variance unbiased estimator [2] does not have a computationally appealing form. This can be addressed as follows at the expense of a slightly less efficient estimator with a considerably reduced bias: we replace the expectations (4) by sums where no pairwise summation indices are identical. This leads to the objective function

$$\frac{1}{m(m-1)} \sum_{i \neq j} K_{ij} L_{ij} + \frac{1}{m^2(m-1)^2} \sum_{i \neq j, u \neq v} K_{ij} L_{uv} - \frac{2}{m(m-1)^2} \sum_{i,j \neq i, v \neq i} K_{ij} L_{iv}. \quad (6)$$

This estimator still has a small degree of bias, albeit significantly reduced since it only arises from the product of expectations over (potentially) independent random variables. Using the shorthand $\tilde{K}_{ij} = K_{ij}(1 - \delta_{ij})$ and $\tilde{L}_{ij} = L_{ij}(1 - \delta_{ij})$ for kernel matrices where the main diagonal terms have been removed we arrive at the expression $(m-1)^{-2} \operatorname{tr} H \tilde{L} H \tilde{K}$. The advantage of this term is that it can be used as a drop-in replacement in Lemma 1.

## 2.3 Mutual Information

An alternative, natural means of studying the dependence between random variables is to compute the mutual information between the random variables $x_i$ and $y_{\pi(i)}$. In general, this is difficult, since it requires density estimation. However, if we assume that $x$ and $y$ are jointly normal in the Reproducing Kernel Hilbert Spaces spanned by the kernels $k, l$ and $k \cdot l$ we can devise an effective approximation of the mutual information. Our reasoning relies on the fact that the differential entropy of a normal distribution with covariance $\Sigma$ is given by

$$h(p) = \tfrac{1}{2} \log |\Sigma| + \text{constant}. \quad (7)$$

Since the mutual information between random variables $X$ and $Y$ is $I(X, Y) = h(X) + h(Y) - h(X, Y)$ we will obtain maximum mutual information by minimizing the joint entropy $h(X, Y)$. Using the Gaussian upper bound on the joint entropy we can maximize a lower bound on the mutual information by minimizing the joint entropy of $J(\pi) := h(X, Y)$. By defining a joint kernel on $\mathfrak{X} \times \mathcal{Y}$ via $k((x, y), (x', y')) = k(x, x')l(y, y')$ we arrive at the optimization problem

$$\operatorname{argmin}_{\pi \in \Pi_m} \log |HJ(\pi)H| \text{ where } J_{ij} = K_{ij} L_{\pi(i), \pi(j)}. \quad (8)$$

Note that this is *related* to the optimization criterion proposed by Jebara [1] in the context of sorting via minimum volume PCA. What we have obtained here is an alternative derivation of Jebara's criterion based on information theoretic considerations. The main difference is that [1] uses the setting to align bags of observations by optimizing $\log |HJ(\pi)H|$ with respect to re-ordering within each of the bags. We will discuss multi-variable alignment at a later stage.

In terms of computation (8) is considerably more expensive to optimize. As we shall see, for the optimization in Lemma 1 a simple iteration over linear assignment problems will lead to desirable solutions, whereas in (8) even computing derivatives is a computational challenge.

## 3  Optimization

**DC Programming**   To find a local maximum of the matching problem we may take recourse to a well-known algorithm, namely DC Programming [4] which in machine learning is also known as the Concave Convex Procedure [5]. It works as follows: for a given function $f(x) = g(x) - h(x)$, where $g$ is convex and $h$ is concave, a lower bound can be found by

$$f(x) \geq g(x_0) + \langle x - x_0, \partial_x g(x_0) \rangle - h(x). \tag{9}$$

This lower bound is convex and it can be maximized effectively over a convex domain. Subsequently one finds a new location $x_0$ and the entire procedure is repeated.

**Lemma 4**  *The function* $\operatorname{tr} \bar{K} \pi^\top \bar{L} \pi$ *is convex in* $\pi$.

Since $\bar{K}, \bar{L} \succeq 0$ we may factorize them as $\bar{K} = U^\top U$ and $\bar{L} = V^\top V$ we may rewrite the objective function as $\left\| V \pi U^\top \right\|^2$ which is clearly a convex quadratic function in $\pi$.

Note that the set of feasible permutations $\pi$ is constrained in a unimodular fashion, that is, the set

$$P_m := \left\{ M \in \mathbb{R}^{m \times m} \text{ where } M_{ij} \geq 0 \text{ and } \sum_i M_{ij} = 1 \text{ and } \sum_j M_{ij} = 1 \right\} \tag{10}$$

has only integral vertices, namely admissible permutation matrices. This means that the following procedure will generate a succession of permutation matrices which will yield a local maximum for the assignment problem:

$$\pi_{i+1} = (1 - \lambda) \pi_i + \lambda \operatorname{argmax}_{\pi \in P_m} \left[ \operatorname{tr} \bar{K} \pi^\top \bar{L} \pi_i \right] \tag{11}$$

Here we may choose $\lambda = 1$ in the last step to ensure integrality. This optimization problem is well known as a Linear Assignment Problem and effective solvers exist for it [6].

**Lemma 5**  *The algorithm described in (11) for* $\lambda = 1$ *terminates in a finite number of steps.*

We know that the objective function may only increase for each step of (11). Moreover, the solution set of the linear assignment problem is finite. Hence the algorithm does not cycle.

**Nonconvex Maximization**   When using the bias corrected version of the objective function the problem is no longer guaranteed to be convex. In this case we need to add a line-search procedure along $\lambda$ which maximizes $\operatorname{tr} H \tilde{K} H [(1 - \lambda) \pi_i + \lambda \hat{\pi}_i]^\top H \tilde{L} H [(1 - \lambda) \pi_i + \lambda \hat{\pi}_i]$. Since the function is quadratic in $\lambda$ we only need to check whether the search direction remains convex in $\lambda$; otherwise we may maximize the term by solving a simple linear equation.

**Initialization**   Since quadratic assignment problems are in general NP hard we may obviously not hope to achieve an optimal solution. That said, a good initialization is critical for good estimation performance. This can be achieved by using Lemma 3. That is, if $\bar{K}$ and $\bar{L}$ only had rank-1, the problem could be solved by sorting $X$ and $Y$ in matching fashion. Instead, we use the projections onto the first principal vectors as initialization in our experiments.

**Relaxation to a constrained eigenvalue problem**   Yet another alternative is to find an approximate solution of the problem in Lemma 1 by solving

$$\operatorname{maximize}_\eta \quad \eta^\top M \eta \text{ subject to } A \eta = b \tag{12}$$

Here the matrix $M = \bar{K} \otimes \bar{L} \in \mathbb{R}^{m^2 \times m^2}$ is given by the outer product of the constituting kernel matrices, $\eta \in \mathbb{R}^{m^2}$ is a vectorized version of the permutation matrix $\pi$, and the constraints imposed by $A$ and $b$ amount to the polytope constraints imposed by $\Pi_m$. This is essentially the approach proposed by [7] in the context of balanced graph matching, albeit with a suboptimal optimization procedure. Instead, one may use the exact algorithm proposed by [8].

The problem with the relaxation (12) is that it does not scale well to large estimation problems (the size of the optimization problem scales $O(m^4)$) and that the relaxation does not guarantee a feasible solution which means that subsequent projection heuristics need to be found. Hence we did not pursue this approach in our experiments.

# 4   Multivariate Extensions

A natural extension is to align several sets of observations. For this purpose we need to introduce a multivariate version of the Hilbert Schmidt Independence Criterion. One way of achieving this goal is to compute the Hilbert Space norm of the difference between the expectation operator for the joint distribution and the expectation operator for the product of the marginal distributions.

Formally, let there be $T$ random variables $x_i \in \mathcal{X}_i$ which are jointly drawn from some distribution $p(x_1, \ldots, x_m)$. Moreover, denote by $k_i : \mathcal{X}_i \times \mathcal{X}_i \to \mathbb{R}$ the corresponding kernels. In this case we can define a kernel on $\mathcal{X}_1 \otimes \ldots \otimes \mathcal{X}_T$ by $k_1 \cdot \ldots k_T$. The expectation operator with respect to the joint distribution and with respect to the product of the marginals is given by [2]

$$\mathbf{E}_{x_1,\ldots,x_T} \left[ \prod_{i=1}^{T} k_i(x_i, \cdot) \right] \quad \text{and} \quad \prod_{i=1}^{T} \mathbf{E}_{x_i} \left[ k_i(x_i, \cdot) \right] \tag{13}$$

respectively. Both terms are equal if and only if all random variables are independent. The squared difference between both is given by

$$\mathbf{E}_{x_{i=1}^T, x'_{i=1}^T} \left[ \prod_{i=1}^{T} k_i(x_i, x'_i) \right] + \prod_{i=1}^{T} \mathbf{E}_{x_i,x'_i}[k_i(x_i, x'_i)] - 2\mathbf{E}_{x_{i=1}^T} \left[ \prod_{i=1}^{T} \mathbf{E}_{x'_i}[k(x_i, x'_i)] \right]. \tag{14}$$

which we refer to as multiway HSIC. A biased empirical estimate of the above is obtained by replacing sums by empirical averages. Denote by $K_i$ the kernel matrix obtained from the kernel $k_i$ on the set of observations $X_i := \{x_{i1}, \ldots, x_{im}\}$. In this case the empirical estimate of (14) is given by

$$\mathrm{HSIC}[X_1, \ldots, X_T] := 1_m^\top \left( \bigodot_{i=1}^{T} K_i \right) 1_m + \prod_{i=1}^{T} 1_m^\top K_i 1_m - 2 \cdot 1_m^\top \left( \bigodot_{i=1}^{T} K_i 1_m \right) \tag{15}$$

where $\odot_{t=1}^{T} *$ denotes elementwise product of its arguments (the '.*' notation of Matlab). To apply this to sorting we only need to define $T$ permutation matrices $\pi_i \in \Pi_m$ and replace the kernel matrices $K_i$ by $\pi_i^\top K_i \pi_i$.

Without loss of generality we may set $\pi_1 = \mathbf{1}$, since we always have the freedom to fix the order of one of the $T$ sets with respect to which the other sets are to be ordered. In terms of optimization the same considerations as presented in Section 3 apply. That is, the objective function is convex in the permutation matrices $\pi_i$ and we may apply DC programming to find a locally optimal solution. The experimental results for multiway HSIC can be found in the appendix.

# 5   Applications

To investigate the performance of our algorithm (it is a fairly nonstandard unsupervised method) we applied it to a variety of different problems ranging from visualization to matching and estimation.

In all our experiments, the maximum number of iterations used in the updates of $\pi$ is 100 and we terminate early if progress is less than $0.001\%$ of the objective function.

## 5.1   Data Visualization

In many cases we may want to visualize data according to the metric structure inherent in it. In particular, we want to align it according to a given template, such as a grid, a torus, or any other *fixed* structure. Such problems occur when presenting images or documents to a user. While there is a large number of algorithms for low dimensional object layout (self organizing maps, maximum variance unfolding, local-linear embedding, generative topographic map, . . . ), most of them suffer from the problem that the low dimensional presentation is nonuniform. This has the advantage of revealing cluster structure but given limited screen size the presentation is undesirable.

Instead, we may use kernelized sorting to align objects. Here the kernel matrix $L$ is given by the similarity measure between the objects $x_i$ that are to be aligned. The kernel $K$, on the other hand, denotes the similarity between the locations where objects are to be aligned to. For the sake of simplicity we used a Gaussian RBF kernel between the objects to laid out and also between the

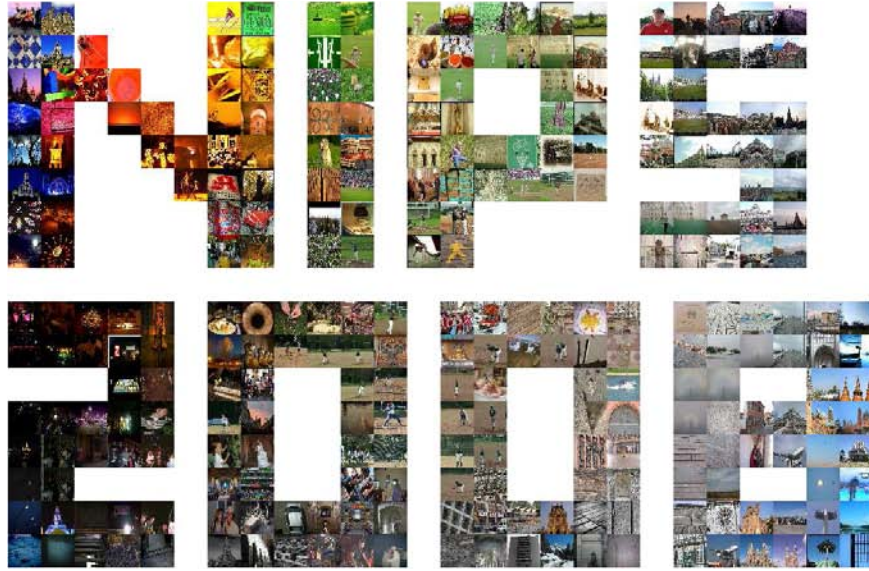

Figure 1: Layout of 284 images into a 'NIPS 2008' letter grid using kernelized sorting.

positions of the grid, i.e. $k(x, x') = \exp(-\gamma \|x - x'\|^2)$. The kernel width $\gamma$ was adjusted to the inverse median of $\|x - x'\|^2$ such that the argument of the exponential is $O(1)$. Our choice of the Gaussian RBF kernel is likely not optimal for the specific set of observations (e.g. SIFT feature extraction followed by a set kernel would be much more appropriate for images). That said we want to emphasize that the gains arise from the *algorithm* rather than a specific choice of a *function class*.

We obtained 284 images from `http://www.flickr.com` which were resized and downsampled to $40 \times 40$ pixels. We converted the images from RGB into Lab color space, yielding $40 \times 40 \times 3$ dimensional objects. The grid, corresponding to $X$ is a 'NIPS 2008' letters on which the images are to be laid out. After sorting we display the images according to their matching coordinates (Figure 1). We can see images with similar color composition are found at proximal locations.

We also lay out the images (we add 36 images to make the number 320) into a 2D grid of $16 \times 20$ mesh using kernelized sorting. For comparison we use a Self-Organizing Map (SOM) and a Generative Topographic Mapping (GTM) and the results are shown in the appendix. Although the images are also arranged according to the color grading, the drawback of SOM (and GTM) is that it creates blank spaces in the layout. This is because SOM maps several images into the same neuron. Hence some neurons may not have data associated with them. While SOM is excellent in grouping similar images together, it falls short in exactly arranging the images into 2D grid.

## 5.2 Matching

To obtain more quantifiable results rather than just generally aesthetically pleasing pictures we apply our algorithm to matching problems where the correct match is known.

**Image matching:** Our first test was to match image halves. For this purpose we used the data from the layout experiment and we cut the images into two $20 \times 40$ pixel patches. The aim was to find an alignment between both halves such that the dependence between them is maximized. In other words, given $x_i$ being the left half of the image and $y_i$ being the right half, we want to find a permutation $\pi$ which lines up $x_i$ and $y_i$.

This would be a trivial undertaking when being able to compare the two image halves $x_i$ and $y_i$. While such comparison is clearly feasible for images where we *know* the compatibility function, it may not be possible for generic objects. The figure is presented in the appendix. For a total of 320 images we recovered 140 pairs. This is quite respectable given that chance level would be 1 correct pair (a random permutation matrix has on expectation one nonzero diagonal entry).

**Estimation** In a next experiment we aim to determine how well the overall quality of the matches is. That is, whether the objects matched share similar properties. For this purpose we used binary, multi-

Table 1: Error rate for matching problems

| Type | Data set | $m$ | Kernelized Sorting | Baseline | Reference |
|------|----------|-----|--------------------|----------|-----------|
| Binary | australian | 690 | 0.29±0.02 | 0.49 | 0.21±0.04 |
| | breastcancer | 683 | 0.06±0.01 | 0.46 | 0.06±0.03 |
| | derm | 358 | 0.08±0.01 | 0.43 | 0.00±0.00 |
| | optdigits | 765 | 0.01±0.00 | 0.49 | 0.01±0.00 |
| | wdbc | 569 | 0.11±0.04 | 0.47 | 0.05±0.02 |
| Multiclass | satimage | 620 | 0.20±0.01 | 0.80 | 0.13±0.04 |
| | segment | 693 | 0.58±0.02 | 0.86 | 0.05±0.02 |
| | vehicle | 423 | 0.58±0.08 | 0.75 | 0.24±0.07 |
| Regression | abalone | 417 | 13.9±1.70 | 18.7 | 6.44±3.14 |
| | bodyfat | 252 | 4.5±0.37 | 7.20 | 3.80±0.76 |

Table 2: Number of correct matches (out of 300) for English aligned documents.

| Source language | Pt | Es | Fr | Sv | Da | It | Nl | De |
|-----------------|-----|-----|-----|-----|-----|-----|-----|-----|
| Kernelized Sorting | 252 | 218 | 246 | 150 | 230 | 237 | 223 | 95 |
| Baseline (length match) | 9 | 12 | 8 | 6 | 6 | 11 | 7 | 4 |
| Reference (dictionary) | 298 | 298 | 298 | 296 | 297 | 300 | 298 | 284 |

class, and regression datasets from the UCI repository `http://archive.ics.uci.edu/ml` and the LibSVM site `http://www.csie.ntu.edu.tw/~cjlin/libsvmtools`.

In our setup we split the dimensions of the data into two sets and permute the data in the second set. The so-generated two datasets are then matched and we use the estimation error to quantify the quality of the match. That is, assume that $y_i$ is associated with the observation $x_i$. In this case we compare $y_i$ and $y_{\pi(i)}$ using binary classification, multiclass, or regression loss accordingly.

To ensure good dependence between the subsets of variables we choose a split which ensures correlation. This is achieved as follows: we pick the dimension with the largest correlation coefficient as a reference. We then choose the coordinates that have at least $0.5$ correlation with the reference and split those equally into two sets, set A and set B. We also split the remainder coordinates equally into the two existing sets and finally put the reference coordinate into set A. This ensures that the set B of dimensions will have strong correlation with at least one dimension in the set A. The listing of the set members for different datasets can be found in the appendix.

The results are summarized in Table 1. As before, we use a Gaussian RBF kernel with median adjustment of the kernel width. To obtain statistically meaningful results we subsample $80\%$ of the data 10 times and compute the error of the match on the subset (this is done in lieu of cross-validation since the latter is meaningless for matching). As baseline we compute the expected performance of random permutations which can be done exactly. Finally, as reference we use SVM classification / regression with results obtained by 10-fold cross-validation. Matching is able to retrieve significant information about the labels of the corresponding classes, in some cases performing as well as a full classification approach.

**Multilingual Document Matching** To illustrate that kernelized sorting is able to recover nontrivial similarity relations we applied our algorithm to the matching of multilingual documents. For this purpose we used the Europarl Parallel Corpus. It is a collection of the proceedings of the European Parliament, dating back to 1996 [9]. We select the 300 longest documents of Danish (Da), Dutch (Nl), English (En), French (Fr), German (De), Italian (It), Portuguese (Pt), Spanish (Es), and Swedish (Sv). The purpose is to match the non-English documents (source languages) to its English translations (target language). Note that our algorithm does *not* require a cross-language dictionary. In fact, one could use kernelized sorting to generate a dictionary after initial matching has occurred.

In keeping with the choice of a simple kernel we used standard TF-IDF (term frequency - inverse document frequency) features of a bag of words kernel. As preprocessing we remove stopwords (via NLTK) and perform stemming using `http://snowball.tartarus.org`. Finally, the feature vectors are normalized to unit length in term of $\ell_2$ norm. Since kernel matrices on documents are notoriously diagonally dominant we use the bias-corrected version of our optimization problem.

As baseline we used a fairly straightforward means of document matching via its length. That is, longer documents in one language will be most probably translated into longer documents in the other language. This observation has also been used in the widely adopted sentence alignment method [10]. As a dictionary-based alternative we translate the documents using Google's translation engine `http://translate.google.com` to find counterparts in the source language. Smallest distance matches in combination with a linear assignment solver are used for the matching.

The experimental results are summarized in Table 2. We describe a line search procedure in Section 3. In practice we find that fixing $\lambda$ at a given step size and choosing the best solution in terms of the objective function for $\lambda \in \{0.1, 0.2, \ldots, 1.0\}$ works better. Further details can be found in the appendix. Low matching performance for the document length-based method might be due to small variance in the document length after we choose the 300 longest documents. The dictionary-based method gives near-to-perfect matching performance. Further in forming the dictionary, we do not perform stemming on English words and thus the dictionary is highly customized to the problem at hand. Our method produces results consistent to the dictionary-based method with notably low performance for matching German documents to its English translations. We conclude that the difficulty of German-English document matching is inherent to this dataset [9]. Arguably the results are quite encouraging as our method uses only a within class similarity measure while still matches more than 2/3 of what is possible by a dictionary-based method.

## 6 Summary and Discussion

In this paper, we generalized sorting by maximizing the dependency between matched pairs or observations by means of the Hilbert Schmidt Independence Criterion. This way we are able to perform matching *without* the need of a cross-domain similarity measure. The proposed sorting algorithm is efficient and it can be applied to a variety of different problems ranging from data visualization to image and multilingual document matching and estimation. Further examples of kernelized sorting and of reference algorithms are given in the appendix.

**Acknowledgments**   NICTA is funded through the Australian Government's *Backing Australia's Ability* initiative, in part through the ARC.This research was supported by the Pascal Network. Parts of this work were done while LS and AJS were working at NICTA.

## References

[1] T. Jebara. Kernelizing sorting, permutation, and alignment for minimum volume PCA. In *Conference on Computational Learning Theory (COLT)*, volume 3120 of *LNAI*, pages 609–623. Springer, 2004.

[2] A.J. Smola, A. Gretton, L. Song, and B. Schölkopf. A hilbert space embedding for distributions. In E. Takimoto, editor, *Algorithmic Learning Theory*, Lecture Notes on Computer Science. Springer, 2007.

[3] K. Fukumizu, F. R. Bach, and M. I. Jordan. Dimensionality reduction for supervised learning with reproducing kernel Hilbert spaces. *J. Mach. Learn. Res.*, 5:73–99, 2004.

[4] T. Pham Dinh and L. Hoai An. A D.C. optimization algorithm for solving the trust-region subproblem. *SIAM Journal on Optimization*, 8(2):476–505, 1988.

[5] A.L. Yuille and A. Rangarajan. The concave-convex procedure. *Neural Computation*, 15:915–936, 2003.

[6] R. Jonker and A. Volgenant. A shortest augmenting path algorithm for dense and sparse linear assignment problems. *Computing*, 38:325–340, 1987.

[7] T. Cour, P. Srinivasan, and J. Shi. Balanced graph matching. In B. Schölkopf, J. Platt, and T. Hofmann, editors, *Advances in Neural Information Processing Systems 19*, pages 313–320. MIT Press, December 2006.

[8] W. Gander, G.H. Golub, and U. von Matt. A constrained eigenvalue problem. In *Linear Algebra Appl. 114-115*, pages 815–839, 1989.

[9] P. Koehn. Europarl: A parallel corpus for statistical machine translation. In *Machine Translation Summit X*, pages 79–86, 2005.

[10] W. A. Gale and K. W. Church. A program for aligning sentences in bilingual corpora. In *Meeting of the Association for Computational Linguistics*, pages 177–184, 1991.